# Inference in continuous-time change-point models

**Florian Stimberg**
Computer Science, TU Berlin
flostim@cs.tu-berlin.de

**Andreas Ruttor**
Computer Science, TU Berlin
ruttor@cs.tu-berlin.de

**Manfred Opper**
Computer Science, TU Berlin
opperm@cs.tu-berlin.de

**Guido Sanguinetti**
School of Informatics, University of Edinburgh
gsanguin@inf.ed.ac.uk

## Abstract

We consider the problem of Bayesian inference for continuous-time multi-stable stochastic systems which can change both their diffusion and drift parameters at discrete times. We propose exact inference and sampling methodologies for two specific cases where the discontinuous dynamics is given by a Poisson process and a two-state Markovian switch. We test the methodology on simulated data, and apply it to two real data sets in finance and systems biology. Our experimental results show that the approach leads to valid inferences and non-trivial insights.

## 1   Introduction

Continuous-time stochastic models play a prominent role in many scientific fields, from biology to physics to economics. While it is often possible to easily simulate from a stochastic model, it is often hard to solve inference or parameter estimation problems, or to assess quantitatively the fit of a model to observations. In recent years this has motivated an increasing interest in the machine learning and statistics community in Bayesian inference approaches for stochastic dynamical systems, with applications ranging from biology [1–3] to genetics [4] to spatio-temporal systems [5].

In this paper, we are interested in modelling and inference for systems exhibiting multi-stable behavior. These systems are characterized by stable periods and rapid transitions between different equilibria. Very common in physical and biological sciences, they are also highly relevant in economics and finance, where unexpected events can trigger sudden changes in trading behavior [6].

While there have been a number of approaches to Bayesian change-point inference [7–9] most of them expect the observations to be independent and coming directly from the change-point process. In many systems this is not the case because observations are only available from a dynamic process whose parameters are change-point processes. There have been other algorithms for detecting indirectly observed change-point processes [10], but we emphasize that we are also (and sometimes mostly) interested in the dynamical parameters of the system.

We present both an exact and an MCMC-based approach for Bayesian inference in multi-stable stochastic systems. We describe in detail two specific scenarios: the classic change-point process scenario whereby the latent process has a new value at each jump and a bistable scenario where the latent process is a stochastic telegraph process. We test extensively our model on simulated data, showing good convergence properties of the sampling algorithm. We then apply our approach to two very diverse data sets in finance and systems biology, demonstrating that the approach leads to valid inferences and interesting insights in the nature of the system.

## 2 The generative model

We consider a system of $N$ stochastic differential equations (SDE)

$$dx_i = (A_i(t) - \lambda_i x_i)dt + \sigma_i(t)dW_i, \tag{1}$$

of the Ornstein-Uhlenbeck type for $i = 1, \ldots, N$, which are driven by independent Wiener processes $W_i(t)$. The time dependencies in the drift $A_i(t)$ and in the diffusion terms $\sigma_i(t)$ will account for sudden changes in the system and will be further modelled by stochastic Markov jump processes. Our prior assumption is that change points, where $A_i$ and $\sigma_i$ change their values, constitute *Poisson events*. This means that the times $\Delta t$ between consecutive change points are independent exponentially distributed random variables with density $p(\Delta t) = f \exp(-f\Delta t)$, where $f$ denotes their expected number per time unit. We will consider two different models for the values of $A_i$ and $\sigma_i$ in this paper:

- **Model 1** assumes that at each of the change points $A_i$ and $\sigma_i$ are drawn independently from fixed prior densities $p_A(\cdot)$ and $p_\sigma(\cdot)$. The number of change points up to time $t$ is counted by the *Poisson process* $\mu(t)$, so that $A_i(t) = A_i^{\mu(t)}$ and $\sigma_i(t) = \sigma_i^{\mu(t)}$ are piecewise constant functions of time.

- **Model 2** restricts the parameters $A_i(t)$ and $\sigma_i(t)$ to two possible values $A_i^0$, $A_i^1$, $\sigma_i^0$, and $\sigma_i^1$, which are time independent random variables with corresponding priors. We select the parameters according to the *telegraph process* $\mu(t)$, which switches between $\mu = 0$ and $\mu = 1$ at each change point.

For both models, $A_i(t)$ and $\sigma_i(t)$ are unobserved. However, we have a data set of $M$ noisy observations $Y \equiv \{\mathbf{y}_1, \ldots, \mathbf{y}_M\}$ of the process $\mathbf{x}(t) = (x_1(t), \ldots, x_N(t))(t)$ at discrete times $t_j$, $j = 1, \ldots, M$, i.e. we assume that $\mathbf{y}_j = \mathbf{x}(t_j) + \boldsymbol{\xi}_j$ with independent Gaussian noise $\boldsymbol{\xi}_j \sim \mathcal{N}(0, \sigma_o^2)$.

## 3 Bayesian Inference

Given data $Y$ we are interested in the posterior distribution of all unobserved quantities, which are the paths of the stochastic processes $X \equiv \mathbf{x}_{[0:T]}$, $Z \equiv (\mathbf{A}_{[0:T]}, \boldsymbol{\sigma}_{[0:T]})$ in a time interval $[0:T]$ and the model parameters $\Lambda = (\{\lambda_i\})$. For simplicity, we have not used a prior for the rate $f$ and treated it as a fixed quantity. The joint probability of these quantities is given by

$$p(Y, X, Z, \Lambda) = p(Y|X)p(X|Z, \Lambda)p(Z)p(\Lambda) \tag{2}$$

A Gibbs sampling approach to this distribution is nontrivial, because the sample paths are infinite dimensional objects, and a naive temporal discretization may lead to potential extra errors.

Inference is greatly facilitated by the fact that *conditioned* on $Z$ and $\Lambda$, $X$ is an Ornstein-Uhlenbeck process, i.e. a Gaussian Markov process. Since also the data likelihood $p(Y|X)$ is Gaussian, it is possible to integrate out the process $X$ *analytically* leading to a marginal posterior

$$p(Z|Y, \Lambda) \propto p(Y|Z, \Lambda)p(Z) \tag{3}$$

over the simpler piecewise constant sample paths of the jump processes. Details on how to compute the likelihood $p(Y|Z, \Lambda)$ are given in the supplementary material.

When inference on posterior values $X$ is required, we can use the fact that $X|Y, Z, \Lambda$ is an inhomogeneous Ornstein-Uhlenbeck process, which allows for an explicit *analytical computation* of marginal means and variances at each time.

The jump processes $Z = \{\boldsymbol{\tau}, \Theta\}$ are completely determined by the set of change points $\boldsymbol{\tau} \equiv \{\tau_j\}$ and the actual values of $\Theta \equiv \{\mathbf{A}^j, \boldsymbol{\sigma}^j\}$ to which the system jumps at the change points. Since $p(Z) = p(\Theta|\boldsymbol{\tau})p(\boldsymbol{\tau})$ and $p(\Theta|\tau, Y, \Lambda) \propto p(Y|Z, \Lambda)p(\Theta|\boldsymbol{\tau})$, we can see that conditioned on a set of, say $m$ change points, the distribution of $\Theta$ is a finite (and usually relatively low) dimensional integral from which one can draw samples using standard methods. In fact, if the prior density of the drift values $p_A$ is a Gaussian, then it is easy to see that also the posterior is Gaussian.

## 4 MCMC sampler architecture

We use a Metropolis-within-Gibbs sampler, which alternates between sampling the parameters $\Lambda$, $\Theta$ from $p(\Lambda|Y, \boldsymbol{\tau}, \Theta)$, $p(\Theta|Y, \boldsymbol{\tau}, \Lambda)$ and the positions $\boldsymbol{\tau}$ of change points from $p(\boldsymbol{\tau}|Y, \Theta, \Lambda)$.

Sampling from $p(\Lambda|Y, \boldsymbol{\tau}, \Theta)$ as well as sampling the $\sigma_i$s from $p(\Theta|Y, \boldsymbol{\tau}, \Lambda)$ is done by a Gaussian random walk Metropolis-Hastings sampler on the logarithm of the parameters, to ensure positivity. Sampling the $A_i$s on the other hand can be done directly if the prior $p(A_i)$ is Gaussian, because then $p(A_i|Y, \boldsymbol{\tau}, \Lambda, \{\sigma_i\})$ is also Gaussian.

Finally, we need to draw change points from their density $p(\boldsymbol{\tau}|Y, \Theta, \Lambda) \propto p(Y|Z, \Lambda)p(\Theta|\boldsymbol{\tau})p(\boldsymbol{\tau})$. Their number $m$ is a random variable with a Poisson prior distribution and for fixed $m$, each $\tau_i$ is uniformly distributed in $[0:T]$. Therefore the prior probability of the sorted list $\tau_1, \ldots, \tau_m$ is given by

$$p(\tau_1, \ldots, \tau_m | f) \propto f^m \, e^{-fT}. \tag{4}$$

For sampling change points we use a Metropolis-Hastings step, which accepts a proposal $\boldsymbol{\tau}^*$ for the positions of the change points with probability

$$A = \min\left(1, \frac{p(\boldsymbol{\tau}^*|Y, \Theta, \Lambda)}{p(\boldsymbol{\tau}|Y, \Theta, \Lambda)} \frac{q(\boldsymbol{\tau}|\boldsymbol{\tau}^*)}{q(\boldsymbol{\tau}^*|\boldsymbol{\tau})}\right), \tag{5}$$

where $q(\boldsymbol{\tau}^*|\boldsymbol{\tau})$ is the proposal probability to generate $\boldsymbol{\tau}^*$ starting from $\boldsymbol{\tau}$. Otherwise the old sample is used again. As proposal for a new $\boldsymbol{\tau}$-path we choose one of three (**model 1**) or five (**model 2**) possible actions, which modify the current sample:

- **Moving a change point**: One change point is chosen at random with equal probability and the new jump time is drawn from a normal distribution with the old jump time as the mean. The normal distribution is truncated at the neighboring jump times to ensure that the order of jump times stays the same.

- **Adding a change point**: We use a uniform distribution over the whole time interval $[0:T]$ to draw the time of the added jump. In case of **model 1** the parameter set $\Theta_i$ for the new interval stays the same and is only changed in the following update of all the $\Theta$ sets. For **model 2** it is randomly decided if the telegraph process $\mu(t)$ is inverted before or after the new change point. This is necessary to allow $\mu$ to change on both ends.

- **Removing a change point**: The change point to remove is chosen at random. For **model 1** the newly joined interval inherits the parameters with equal probability from the interval before or after the removed change point. As for adding a change point, when using **model 2** we choose to either invert $\mu$ after or before the removed jump time.

For **model 2** we also need the option to add or remove two jumps, because adding or removing one jump will result in inverting the whole process after or before it, which leads to poor acceptance rates. When adding or removing two jumps instead, $\mu$ only changes between these two jumps.

- **Adding two change points**: The first change point is drawn as for adding a single one, the second one is drawn uniformly from the interval between the new and the next change point.

- **Removing two change points**: We choose one of the change points, except the last one, at random and delete it along with the following one.

While the proposal does not use any information from the data, it is very fast to compute and quickly converges to reasonable states, although we initialize the change points simply by drawing from $p(\boldsymbol{\tau})$.

## 5 Exact inference

In the case of small systems described by **model 2** it is also feasible to calculate the marginal probability distribution $q(\mu, x, t)$ for the state variables $x, \mu$ at time $t$ of the posterior process directly. For that purpose, we use a smoothing algorithm, which is quite similar to the well-known method for state inference in hidden Markov models. In order to improve clarity we only discuss the case of

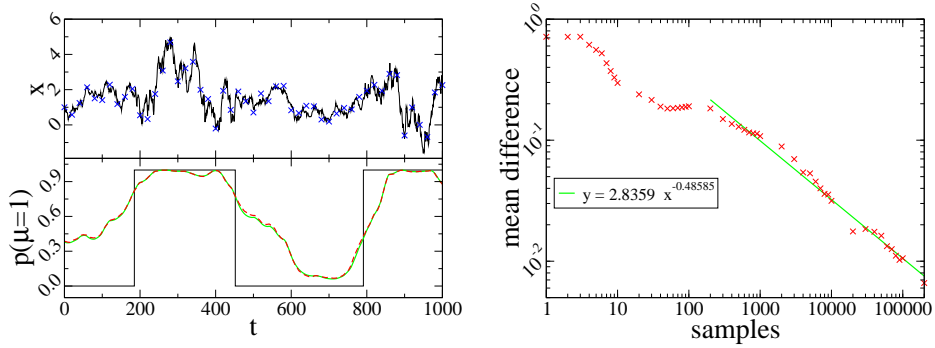

Figure 1: Comparison of the results of the MCMC sampler and the exact inference: (*top left*) True path of $x$ (black) and the noisy observations (blue crosses). (*bottom left*) True path of $\mu$ (black) and posterior of $p(\mu = 1)$ from the exact inference (green) and the MCMC sampler (red dashed). (*right*) Convergence of the sampler. Mean difference between sampler result and exact inference of $p(\mu = 1)$ for different number of samples (red crosses) and the result of power law regression for more than 100 samples (green).

a one-dimensional Ornstein-Uhlenbeck process $x(t)$ here, but the generalization to multiple dimensions is straightforward.

As our model has the Markov property, the exact marginal posterior is given by

$$q(\mu, x, t) = \frac{1}{L} p(\mu, x, t) \psi(\mu, x, t).$$ (6)

Here $p(\mu, x, t)$ denotes the marginal filtering distribution, which is the probability density of the state $(x, \mu)$ at time $t$ conditioned on the observations up to time $t$. The normalization constant $L$ is equal to the total likelihood of all observations. And the last factor $\psi(\mu, x, t)$ is the likelihood of the observations after time $t$ under the condition that the process started with state $(x, \mu)$ at time $t$.

The initial condition for the *forward message* $p(\mu, x, t)$ is the prior over the initial state of the system. The time evolution of the forward message is given by the *forward Chapman-Kolmogorov* equation

$$\left[ \frac{\partial}{\partial t} + \frac{\partial}{\partial x}(A_\mu - \lambda x) - \frac{\sigma_\mu^2}{2} \frac{\partial^2}{\partial x^2} \right] p(\mu, x, t) = \sum_{\nu \neq \mu} \left[ f_{\nu \to \mu} \, p(\nu, x, t) - f_{\mu \to \nu} \, p(\mu, x, t) \right].$$ (7)

Here $f_{\nu \to \mu}$ denotes the transition rate from discrete state $\nu$ to discrete state $\mu \in \{0, 1\}$ of **model 2**, which has the values

$$f_{0 \to 1} = f_{1 \to 0} = f, f_{0 \to 0} = f_{0 \to 0} = 0.$$ (8)

Including an observation $y_j$ at time $t_j$ leads to a jump of the filtering distribution,

$$p(\mu, x, t_j^+) = p(\mu, x, t_j^-) p(y_j | x),$$ (9)

where $p(y_j | x)$ denotes the local likelihood of that observation given by the noise model and $p(\mu, x, t_j^{\mp})$ are the values of the forward message directly before and after time point $t_j$. By integrating equation (7) forward in time from the first observation to the last, we obtain the exact solution to the filtering problem of our model.

Similarly we integrate backward in time from the last observation at time $T$ to the first one in order to compute $\psi(\mu, x, t)$. The initial condition here is $\psi(\mu, x, t_N^+) = 1$. Between observations the time evolution of the backward message is given by the *backward Chapman-Kolmogorov equation*

$$\left[ \frac{\partial}{\partial t} + (A_\mu - \lambda x) \frac{\partial}{\partial x} + \frac{\sigma_\mu^2}{2} \frac{\partial^2}{\partial x^2} \right] \psi(\mu, x, t) = \sum_{\nu \neq \mu} f_{\mu \to \nu} \left[ \psi(\mu, x, t) - \psi(\nu, x, t) \right].$$ (10)

And each observation is taken into account by the jump condition

$$\psi(\mu, x, t_j^-) = \psi(\mu, x, t_j^+) p(y_j | x(t_j)).$$ (11)

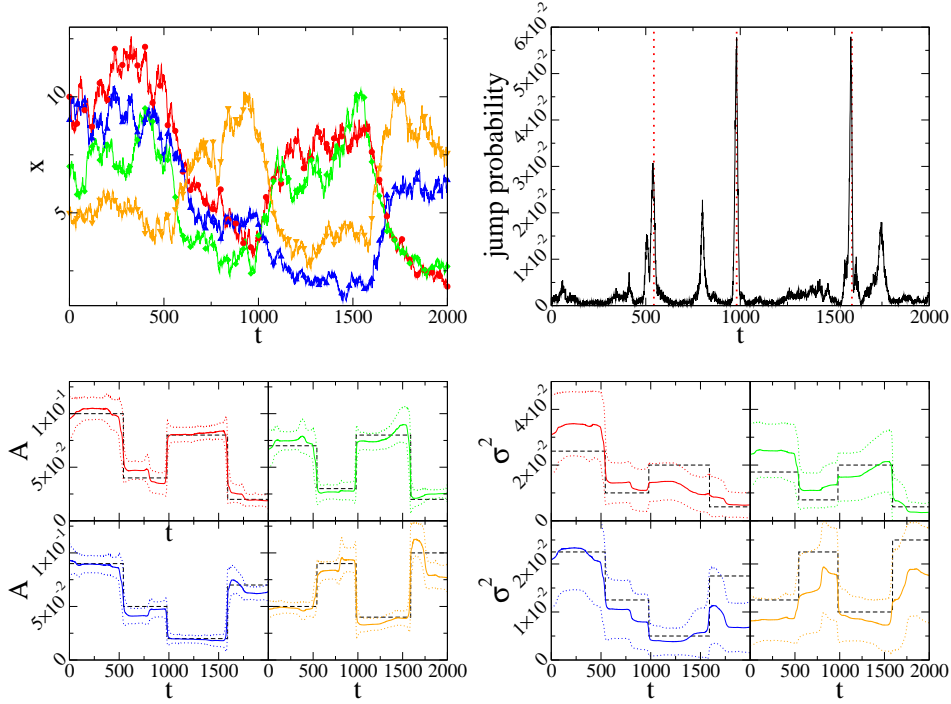

Figure 2: Synthetic results on a four-dimensional diffusion process with diagonal diffusion matrix: (*top left*) true paths with subsampled data points (dots); (*top right*) intensity of the posterior point process (the probability of a change point in a given interval is given by the integral of the intensity). Actual change points are shown as vertical dotted lines. (*bottom row*) posterior processes for $A$ (*left*) and $\sigma^2$ (*right*) with a one standard deviation confidence interval. True paths are shown as black dashed lines.

Afterwards, $Lq(\mu, x, t)$ can be calculated by multiplying forward message $p(\mu, x, t)$ and backward message $\psi(\mu, x, t)$. Normalizing that quantity according to

$$\int \sum_{\mu} q(\mu, x, t) dx = 1 \tag{12}$$

then gives us the marginal posterior as well as the total likelihood $L = p(y_1, \ldots, y_N | A, b, \ldots)$ of all observations. Note, that we only need to calculate $L$ for one time point, as it is a time-independent quantity. Minimizing $-\log L$ as a function of the parameters can then be used to obtain maximum likelihood estimates. As an analytical solution for both equations (7) and (10) does not exist, we have to integrate them numerically on a grid. A detailed description is given in the supplementary material.

## 6 Results

### 6.1 Synthetic Data

As a first consistency check, we tested the model on simulated data. The availability of an exact solution to the inference problem provides us with an excellent way of monitoring convergence of our sampler. Figure 1 shows the results of sampling on data generated from **model 2**, with parameter settings such that only the diffusion constant changes, making it a fairly challenging problem. Despite the rather noisy nature of the data (top left panel), the approach gives a reasonable reconstruction of the latent switching process (left panel, bottom). The comparison between exact inference and MCMC is also instructive, showing that the sampled posterior does indeed converge to the true posterior after a relatively short burn in period (Figure 1 right panel). A power law regression of the mean absolute difference between exact and MCMC (after burn in) on the number

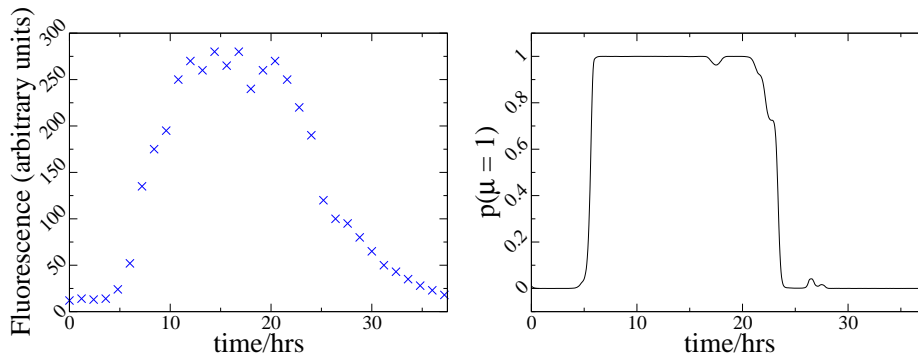

Figure 3: Stochastic gene expression during competence: (*left*) fluorescence intensity for comS protein over 36 hrs; (*right*) inferred comK activation profile using model 2 (see text)

of samples yields a decrease with approximately the square root of the number of samples (exponent 0.48), as expected.

To test the performance of the inference approach on **model 1**, we simulated data from a four-dimensional diffusion process with diagonal diffusion with change points in the drift and diffusion (at the same times). The results of the sampling based inference are shown in Figure 2. Once again, the results indicate that the sampled distribution was able to accurately identify the change points (top right panel) and the values of the parameters (bottom panels). The results are based on 260,000 samples and were obtained in approximately twelve hours on a standard workstation. Unfortunately in this higher dimensional example we do not have access to the true posterior, as numerical integration of a high dimensional PDE proved computationally prohibitive.

## 6.2 Characterization of noise in stochastic gene expression

Recent developments in microscopy technology have led to the startling discovery that stochasticity plays a crucial role in biology [11]. A particularly interesting development is the distinction between *intrinsic* and *extrinsic* noise [12]: given a biological system, intrinsic noise arises as a consequence of fluctuations due to the low numbers of the molecular species composing the system, while extrinsic noise is caused by external changes influencing the system of interest. A currently open question is how to characterize mathematically the difference between intrinsic and extrinsic noise, and a widely mooted opinion is that either the amplitude or the spectral characteristics of the two types of noise should be different [13]. To provide a proof-of-principle investigation into these issues, we tested our model on real stochastic gene expression data subject to extrinsic noise in *Bacillus subtilis* [14]. Here, single-cell fluorescence levels of the protein comS were assayed through time-lapse microscopy over a period of 36 hours. During this period, the protein was subjected to extrinsic noise in the form of activation of the regulator comK, which controls comS expression with a switch-like behavior (Hill coefficient 5). Activation of comS produces a striking phenotype called *competence*, whereby the cell stops dividing, becoming visibly much longer than sister cells. The data used is shown in Figure 3, left panel.

To determine whether the noise characteristics are different in the presence of comK activity, we modelled the data using two different models: **model 2**, where both the offset $A$ and the diffusion $\sigma$ can take two different values, and a constrained version of **model 2** where the diffusion constant cannot switch (as in [15]). In both cases we sampled 500,000 posterior samples, discarding an initial burn-in of 10,000 samples. Both models predict two clear change points representing the activation and inactivation of comK at approximately 5 and 23 hrs respectively (Figure 3 right panel, showing **model 2** results). Also both models are in close agreement on the inferred kinetic parameters $A$, $b$, and $\lambda$ (Figure 4, left panel, showing a comparison of the $\lambda$ posteriors), consistently with the fact that the mean trajectory for both models must be the same.

Naturally, **model 2** predicted two different values for the diffusion constant depending on the activity state of comK (Figure 4, central panel). The two posterior distributions for $\sigma_1$ and $\sigma_2$ appear to be well separated, lending support to the unconstrained version of **model 2** being a better description

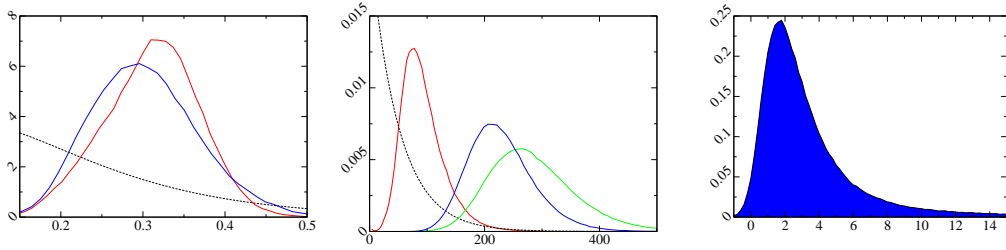

Figure 4: Stochastic gene expression during competence: (*left*) posterior estimates of $\lambda$ (solid) for switching $\sigma$ (red) and non-switching $\sigma^2$ (blue) with common prior (dashed); (*center*) posterior estimates of $\sigma_1^2$ (red solid), $\sigma_2^2$ (green solid) and non-switching $\sigma$ posterior (blue solid) with common prior (dashed); (*right*) posterior distribution of $f(A, b, \sigma_1, \sigma_2)$ (see text), indicating the incompatibility of the simple birth-death model of steady state with the data.

of the data. While this is an interesting result in itself, it is perhaps not surprising. We can gain some insights by considering the underlying discrete dynamics of comS protein counts, which our model approximates as a continuous variable [16]. As we are dealing with bacterial cells, transcription and translation are tightly coupled, so that we can reasonably assume that protein production is given by a Poisson process. At steady state in the absence of comK, the production of comS proteins will be given by a birth-death process with birth rate $b$ and death rate $\lambda$, while in the presence of comK the birth rate would change to $A + b$. Defining

$$\rho_0 = \frac{b}{\lambda}, \qquad \rho_1 = \frac{A + b}{\lambda} \tag{13}$$

this simple birth-death model implies a Poisson distribution of the steady state comS protein levels in the two comK states, with parameters $\rho_0$, $\rho_1$ respectively. Unfortunately, we only measure the counts of comS protein up to a proportionality constant (due to the arbitrary units of fluorescence); this means that the basic property of Poisson distributions of having the same mean and variance cannot be tested easily. However, if we consider the ratio of signal to noise ratios in the two states, we obtain a quantity which is independent of the fluorescence units, namely

$$\frac{\bar{N}_1/\text{stdev}(N_1)}{\bar{N}_0/\text{stdev}(N_0)} = \sqrt{\frac{\rho_1}{\rho_0}} = \sqrt{\frac{A + b}{b}}. \tag{14}$$

This relationship is not enforced in our model, but, if the simple birth-death interpretation is supported by the data, it should emerge naturally in the posterior distributions. To test this, we plot in Figure 4 right panel the posterior distribution of

$$f(A, b, \sigma_1, \sigma_2) = \frac{(A + b)/\sigma_2}{b/\sigma_1} - \sqrt{\frac{A + b}{b}}, \tag{15}$$

the difference between the posterior estimate of the ratio of the signal to noise ratios in the two comK states and the prediction from the birth-death model. The overwhelming majority of the posterior probability mass is away from zero, indicating that the data does not support the predictions of the birth-death interpretation of the steady states. A possible explanation of this unexpected result is that the continuous approximation breaks down in the low abundance state (corresponding to no comK activation); the expected number of particles in the comK inactive state is given by $\rho_0$ and has posterior mean 25.8. The breaking down of the OU approximation for these levels of protein expression would be surprising, and would sound a call for caution when using SDEs to model single cell data as advocated in large parts of the literature [2]. An alternative and biologically more exciting explanation would be that the assumption that the decay rates are the same irrespective of the activity of comK is wrong. Notice that, if we assumed different decay rates in the two states, the first term in equation (15) would not change, while the second would scale with a factor $\sqrt{\lambda_0/\lambda_1}$. Our results would then predict that comK regulation at the transcriptional level alone cannot explain the data, and that comS dynamics must be regulated both transcriptionally and post-transcriptionally.

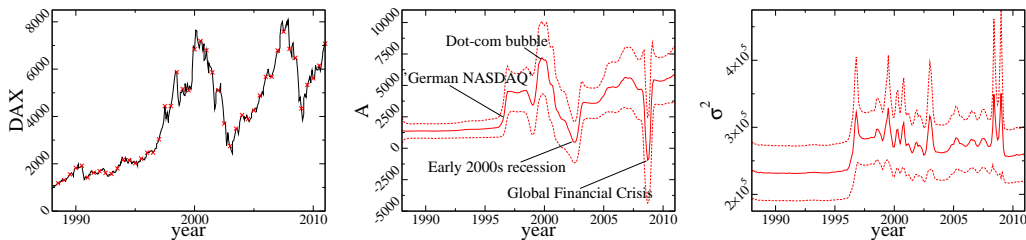

Figure 5: Analysis of DAX data: *left* monthly closing values with data points (red crosses); *center* $A$ process with notable events highlighted; *right* $\sigma$ process.

### 6.3 Change point detection in financial data

As an example of another application of our methodology, we applied **model 1** to financial data taken from the German stock exchange (DAX). The data, shown in Figure 5, consists of monthly closing values; we subsampled it at quarterly values. The posterior processes for $A$ and $\sigma$ are shown in the central and right panels of Figure 5 respectively. An inspection of these results reveals several interesting change points which can be related to known events: for convenience, we highlight a few of them in the central panel of Figure 5. Clearly evident are the changes caused by the introduction of the *Neuer Markt* (the German equivalent of the NASDAQ) in 1997, as well as the dot-com bubble (and subsequent recession) in the early 2000s and the global financial crisis in 2008. Interestingly, in our results the diffusion (or volatility as is more commonly termed in financial modelling) seems not to be particularly affected by recent events (after surging for the Neuer Markt). A possible explanation is the rather long time interval between data points: volatility is expected to be particularly high on the micro-time scale, or at best the daily scale. Therefore the effective sampling rate we use may be too sparse to capture these changes.

## 7 Discussion

In this paper, we proposed a Bayesian approach to inference in multi-stable system. The basic model is a system of SDEs whose drift and diffusion coefficients can change abruptly at random, exponential distributed times. We describe the approach in two special models: a system of SDEs with coefficients changing at change points from a Poisson process (**model 1**) and a system of SDE whose coefficients can change between two sets of values according to a random telegraph process (**model 2**). Each model is particularly suitable for specific applications: while **model 1** is important in financial modelling and industrial application, **model 2** extends a number of similar models already employed in systems biology [3,15,17]. Testing our model(s) in specific applications reveals that it often leads to interpretable predictions. For example, in the analysis of DAX data, the model correctly captures known important events such as the dot-com bubble. In an application to biological data, the model leads to non-obvious predictions of considerable biological interest.

In regard to the computational costs stated in this paper, it has to be noted that the sampler was implemented in Matlab. A new implementation in C++ for **model 2** showed over 12 times faster computational times for a data set with 10 OU processes and 2 telegraph processes. A similar improvement is to be expected for **model 1**.

There are several interesting possible avenues to further this work. While the inference scheme we propose is practical in many situations, scaling to higher dimensional problems may become computationally intensive. It would therefore be interesting to investigate approximate inference solutions like the ones presented in [15]. Another interesting direction would be to extend the current work to a factorial design; these can be important, particularly in biological applications where multiple factors can interact in determining gene expression [17, 18]. Finally, our models are naturally non-parametric in the sense that the number of change points is not a priori determined. It would be interesting to explore further non-parametric extensions where the system can exist in a finite but unknown number of regimes, in the spirit of non-parametric models for discrete time dynamical systems [19].

## References

[1] Neil D. Lawrence, Guido Sanguinetti, and Magnus Rattray. Modelling transcriptional regulation using Gaussian processes. In B. Schölkopf, J. Platt, and T. Hoffman, editors, *Advances in Neural Information Processing Systems 19*. 2007.

[2] Darren J. Wilkinson. *Stochastic Modelling for Systems Biology*. Chapman & Hall / CRC, London, 2006.

[3] Guido Sanguinetti, Andreas Ruttor, Manfred Opper, and Cèdric Archambeau. Switching regulatory models of cellular stress response. *Bioinformatics*, 25:1280–1286, 2009.

[4] Ido Cohn, Tal El-Hay, Nir Friedman, and Raz Kupferman. Mean field variational approximation for continuous-time Bayesian networks. In *Proceedings of the twenty-fifthth conference on Uncertainty in Artificial Intelligence (UAI)*, 2009.

[5] Andreas Ruttor and Manfred Opper. Approximate inference in reaction-diffusion processes. *JMLR W&CP*, 9:669–676, 2010.

[6] Tobias Preis, Johannes Schneider, and H. Eugene Stanley. Switching processes in financial markets. *Proceedings of the National Academy of Sciences USA*, 108(19):7674–7678, 2011.

[7] Paul Fearnhead and Zhen Liu. Efficient bayesian analysis of multiple changepoint models with dependence across segments. *Statistics and Computing*, 21(2):217–229, 2011.

[8] Paolo Giordani and Robert Kohn. Efficient bayesian inference for multiple change-point and mixture innovation models. *Journal of Business and Economic Statistics*, 26(1):66–77, 2008.

[9] E. B. Fox, E. B. Sudderth, M. I. Jordan, and A. S. Willsky. An HDP-HMM for systems with state persistence. In *Proc. International Conference on Machine Learning*, July 2008.

[10] Yunus Saatci, Ryan Turner, and Carl Edward Rasmussen. Gaussian process change point models. In *ICML*, pages 927–934, 2010.

[11] Vahid Shahrezaei and Peter Swain. The stochastic nature of biochemical networks. *Curr. Opin. in Biotech.*, 19(4):369–374, 2008.

[12] Michael B. Elowitz, Arnold J. Levine, Eric D. Siggia, and Peter S. Swain. Stochastic gene expression in a single cell. *Science*, 297(5584):1129–1131, 2002.

[13] Avigdor Eldar and Michael B. Elowitz. Functional roles for noise in genetic circuits. *Nature*, 467(7312):167–173, 2010.

[14] Gürol M. Suël, Jordi Garcia-Ojalvo, Louisa M. Liberman, and Michael B. Elowitz. An excitable gene regulatory circuit induces transient cellular differentiation. *Nature*, 440(7083):545–550, 2006.

[15] Manfred Opper, Andreas Ruttor, and Guido Sanguinetti. Approximate inference in continuous time gaussian-jump processes. In J. Lafferty, C. K. I. Williams, R. Zemel, J. Shawe-Taylor, and A. Culotta, editors, *Advances in Neural Information Processing Systems 23*, pages 1822–1830. 2010.

[16] N. G. van Kampen. *Stochastic Processes in Physics and Chemistry*. North-Holland, Amsterdam, 1981.

[17] Manfred Opper and Guido Sanguinetti. Learning combinatorial transcriptional dynamics from gene expression data. *Bioinformatics*, 26(13):1623–1629, 2010.

[18] H. M. Shahzad Asif and Guido Sanguinetti. Large scale learning of combinatorial transcriptional dynamics from gene expression. *Bioinformatics*, 27(9):1277–1283, 2011.

[19] Matthew Beal, Zoubin Ghahramani, and Carl Edward Rasmussen. The infinite hidden Markov model. In S. Becker, S. Thrun, and L. Saul, editors, *Advances in Neural Information Processing Systems 14*, pages 577–584. 2002.

